# Similarity by Composition

**Oren Boiman**     **Michal Irani**
Dept. of Computer Science and Applied Math
The Weizmann Institute of Science
76100 Rehovot, Israel

## Abstract

We propose a new approach for measuring similarity between two signals, which is applicable to many machine learning tasks, and to many signal types. We say that a signal $S_1$ is "similar" to a signal $S_2$ if it is "easy" to compose $S_1$ from few *large* contiguous chunks of $S_2$. Obviously, if we use small enough pieces, then any signal can be composed of any other. Therefore, the larger those pieces are, the more similar $S_1$ is to $S_2$. This induces a *local similarity score at every point* in the signal, based on the size of its supported surrounding region. These local scores can in turn be accumulated in a principled information-theoretic way into a *global similarity score* of the entire $S_1$ to $S_2$. "Similarity by Composition" can be applied between *pairs of signals*, between *groups of signals*, and also *between different portions of the same signal*. It can therefore be employed in a wide variety of machine learning problems (clustering, classification, retrieval, segmentation, attention, saliency, labelling, etc.), and can be applied to a wide range of signal types (images, video, audio, biological data, etc.) We show a few such examples.

## 1  Introduction

A good measure for similarity between signals is necessary in many machine learning problems. However, the notion of "similarity" between signals can be quite complex. For example, observing Fig. 1, one would probably agree that Image-B is more "similar" to Image-A than Image-C is. But why...? The configurations appearing in image-B are different than the ones observed in Image-A. What is it that makes those two images more similar than Image-C? Commonly used similarity measures would not be able to detect this type of similarity. For example, standard global similarity measures (e.g., Mutual Information [12], Correlation, SSD, etc.) require prior alignment or prior knowledge of dense correspondences between signals, and are therefore not applicable here. Distance measures that are based on comparing empirical distributions of local features, such as "bags of features" (e.g., [11]), will not suffice either, since all three images contain similar types of local features (and therefore Image-C will also be determined similar to Image-A).

In this paper we present a new notion of similarity between signals, and demonstrate its applicability to several machine learning problems and to several signal types. Observing the right side of Fig. 1, it is evident that Image-B can be composed relatively easily from few large chunks of Image-A (see color-coded regions). Obviously, if we use small enough pieces, then any signal can be composed of any other (including Image-C from Image-A). We would like to employ this idea to indicate high similarity of Image-B to Image-A, and lower similarity of Image-C to Image-A. In other words, regions in one signal (the "query" signal) which can be composed using *large contiguous chunks of data* from the other signal (the "reference" signal) are considered to have high local similarity. On the other hand, regions in the query signal which can be composed only by using small fragmented pieces are considered locally **dis**similar. This induces a *similarity score at every point in the signal* based on the size of its largest surrounding region which can be found in the other signal (allowing for some distortions). This approach provides the ability to generalize and infer about new configurations in the query signal that were never observed in the reference signal, while preserving

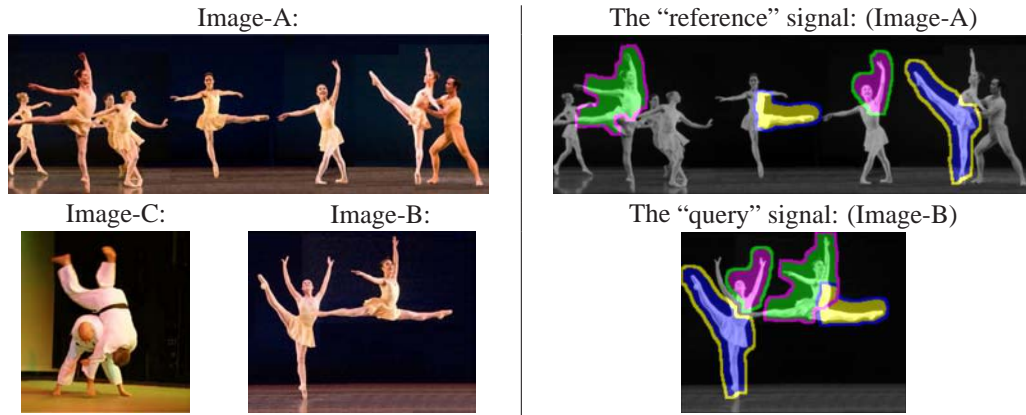

Image-A:

The "reference" signal: (Image-A)

Image-C:    Image-B:

The "query" signal: (Image-B)

Figure 1: **Inference by Composition – Basic concept.**
**Left:** *What makes "Image-B" look more similar to "Image-A" than "Image-C" does?* (*None of the ballet configurations in "Image-B" appear in "Image-A"!*)
**Right:** *Image-B (the "query") can be composed using few large contiguous chunks from Image-A (the "reference"), whereas it is more difficult to compose Image-C this way. The large shared regions between B and A (indicated by colors) provide high evidence to their similarity.*

structural information. For instance, even though the two ballet configurations observed in Image-B (the "query" signal) were never observed in Image-A (the "reference" signal), they can be inferred from Image-A via composition (see Fig. 1), whereas the configurations in Image-C are much harder to compose.

Note that the shared regions between similar signals are typically *irregularly shaped*, and therefore cannot be restricted to predefined regularly shaped partitioning of the signal. The shapes of those regions are data dependent, and cannot be predefined. Our notion of signal composition is "geometric" and data-driven. In that sense it is very different from standard decomposition methods (e.g., PCA, ICA, wavelets, etc.) which seek linear decomposition of the signal, but not geometric decomposition. Other attempts to maintain the benefits of local similarity while maintaining global structural information have recently been proposed [8]. These have been shown to improve upon simple "bags of features", but are restricted to preselected partitioning of the image into rectangular sub-regions.

In our previous work [5] we presented an approach for detecting irregularities in images/video as regions that cannot be composed from large pieces of data from other images/video. Our approach was restricted only to detecting local irregularities. In this paper we extend this approach to a general principled theory of "Similarity by Composition", from which we derive local and global similarity and dissimilarity measures between signals. We further show that this framework extends to a wider range of machine learning problems and to a wider variety of signals ($1D, 2D, 3D, ..$ signals).

More formally, we present a statistical (generative) model for composing one signal from another. Using this model we derive information-theoretic measures for local and global similarities induced by shared regions. The local similarities of shared regions ("local evidence scores") are accumulated into a global similarity score ("global evidence score") of the entire query signal relative to the reference signal. We further prove upper and lower bounds on the global evidence score, which are computationally tractable. We present both a theoretical and an algorithmic framework to compute, accumulate and weight those gathered "pieces of evidence".

Similarity-by-Composition is not restricted to pairs of signals. It can also be applied to compute similarity of a signal to a *group of signals* (i.e., compose a query signal from pieces extracted from multiple reference signals). Similarly, it can be applied to measure similarity between two different groups of signals. Thus, Similarity-by-Composition is suitable for detection, retrieval, classification, and clustering. Moreover, it can also be used for measuring similarity or dissimilarity *between different portions of the same signal*. Intra-signal dissimilarities can be used for detecting irregularities or saliency, while intra-signal similarities can be used as affinity measures for sophisticated intra-signal clustering and segmentation.

The importance of large shared regions between signals have been recognized by biologists for determining similarities between DNA sequences, amino acid chains, etc. Tools for finding large repetitions in biological data have been developed (e.g., "BLAST" [1]). In principle, results of such tools can be fed into our theoretical framework, to obtain similarity scores between biological data sequences in a principled information theoretic way.

The rest of the paper is organized as follows: In Sec. 2 we derive information-theoretic measures for local and global "evidence" (similarity) induced by shared regions. Sec. 3 describes an algorithmic framework for computing those measures. Sec. 4 demonstrates the applicability of the derived local and global similarity measures for various machine learning tasks and several types of signal.

## 2  Similarity by Composition – Theoretical Framework

We derive principled information-theoretic measures for local and global similarity between a "query" $Q$ (one or more signals) and a "reference" $ref$ (one or more signals). Large shared regions between $Q$ and $ref$ provide high statistical evidence to their similarity. In this section we show how to quantify this statistical evidence. We first formulate the notion of "local evidence" for local regions within $Q$ (Sec. 2.1). We then show how these pieces of local evidence can be integrated to provide "global evidence" for the entire query $Q$ (Sec. 2.2).

### 2.1  Local Evidence

Let $R \subseteq Q$ be a connected region within $Q$. Assume that a similar region exists in $ref$. We would like to quantify the statistical significance of this region co-occurrence, and show that it increases with the size of $R$. To do so, we will compare the likelihood that $R$ was generated by $ref$, versus the likelihood that it was generated by some random process.

More formally, we denote by $H_{ref}$ the hypothesis that $R$ was "generated" by $ref$, and by $H_0$ the hypothesis that $R$ was generated by a random process, or by any other application-dependent PDF (referred to as the "null hypothesis").

$H_{ref}$ assumes the following model for the "generation" of $R$: a region was taken from somewhere in $ref$, was globally transformed by some global transformation $T$, followed by some small possible local distortions, and then put into $Q$ to generate $R$. $T$ can account for shifts, scaling, rotations, etc. In the simplest case (only shifts), $T$ is the corresponding location in $ref$.

We can compute the likelihood ratio:
$$LR(R) = \frac{P(R|H_{ref})}{P(R|H_0)} = \frac{\sum_{T} P(R|T, H_{ref})P(T|H_{ref})}{P(R|H_0)} \qquad (1)$$

where $P(T|H_{ref})$ is the prior probability on the global transformations $T$ (shifts, scaling, rotations), and $P(R|T, H_{ref})$ is the likelihood that $R$ was generated from $ref$ at that location, scale, etc. (up to some local distortions which are also modelled by $P(R|T, H_{ref})$ – see algorithmic details in Sec. 3). If there are multiple corresponding regions in $ref$, (i.e., multiple $T$s), all of them contribute to the estimation of $LR(R)$. We define the *Local Evidence Score* of $R$ to be the log likelihood ratio:

$$LES(R|H_{ref}) = \log_2(LR(R)).$$

$LES$ is referred to as a "local evidence score", because the higher $LES$ is, the smaller the probability that $R$ was generated by random ($H_0$). In fact, $P(\ LES(R|H_{ref}) > l\ |\ H_0) < 2^{-l}$, i.e., the probability of getting a score $LES(R) > l$ for a randomly generated region $R$ is smaller than $2^{-l}$ (this is due to $LES$ being a log-likelihood ratio [3]). High $LES$ therefore provides higher statistical evidence that $R$ was generated from $ref$.

Note that the larger the region $R \subseteq Q$ is, the higher its evidence score $LES(R|H_{ref})$ (and therefore it will also provide higher statistical evidence to the hypothesis that $Q$ was composed from $ref$). For example, assume for simplicity that $R$ has a single identical copy in $ref$, and that $T$ is restricted to shifts with uniform probability (i.e., $P(T|H_{ref}) = const$), then $P(R|H_{ref})$ is constant, regardless of the size of $R$. On the other hand, $P(R|H_0)$ decreases exponentially with the size of $R$. Therefore, the likelihood ratio of $R$ increases, and so does its evidence score $LES$.

$LES$ can also be interpreted as the number of bits *saved* by describing the region $R$ using $ref$, instead of describing it using $H_0$: Recall that the optimal average code length of a random variable $y$ with probability function $P(y)$ is $length(y) = -log(P(y))$. Therefore we can write the evidence score as $LES(R|H_{ref}) = length(R|H_0) - length(R|H_{ref})$. Therefore, larger regions provide higher saving (in bits) in the description length of $R$.

A region $R$ induces "average savings per point" for every point $q \in R$, namely, $\frac{LES(R|H_{ref})}{|R|}$ (where $|R|$ is the number of points in $R$). However, a point $q \in R$ may also be contained in other regions generated by $ref$, each with its own local evidence score. We can therefore define the *maximal possible savings per point* (which we will refer to in short as $PES$ = "Point Evidence Score"):

$$PES(q|H_{ref}) = \max_{R \subseteq Q \ s.t. \ q \in R} \frac{LES(R|H_{ref})}{|R|} \tag{2}$$

For any point $q \in Q$ we define $R_{[q]}$ to be the region which provides this maximal score for $q$. Fig. 1 shows such maximal regions found in Image-B (the query $Q$) given Image-A (the reference $ref$). In practice, many points share the same maximal region. Computing an approximation of $LES(R|H_{ref})$, $PES(q|H_{ref})$, and $R_{[q]}$ can be done efficiently (see Sec 3).

## 2.2 Global Evidence

We now proceed to accumulate multiple local pieces of evidence. Let $R_1, ..., R_k \subseteq Q$ be $k$ disjoint regions in $Q$, which have been generated independently from the examples in $ref$. Let $R_0 = Q \setminus \cup_{i=1}^{k} R_i$ denote the remainder of $Q$. Namely, $S = \{R_0, R_1, ..., R_k\}$ is a segmentation/division of $Q$. Assuming that the remainder $R_0$ was generated i.i.d. by the null hypothesis $H_0$, we can derive a *global evidence score* on the hypothesis that $Q$ was generated from $ref$ via the segmentation $S$ (for simplicity of notation we use the symbol $H_{ref}$ also to denote the global hypothesis):

$$GES(Q|H_{ref}, S) = \log \frac{P(Q|H_{ref}, S)}{P(Q|H_0)} = \log \frac{P(R_0|H_0) \prod_{i=1}^{k} P(R_i|H_{ref})}{\prod_{i=0}^{k} P(R_i|H_0)} = \sum_{i=1}^{k} LES(R_i|H_{ref})$$

Namely, the global evidence induced by $S$ is the accumulated sum of the local evidences provided by the individual segments of $S$. The statistical significance of such an accumulated evidence is expressed by: $P(\ GES(Q|H_{ref}, S) > l \ | \ H_0) = P(\ \sum_{i=1}^{k} LES(R_i|H_{ref}) > l \ | \ H_0) < 2^{-l}$.

Consequently, we can accumulate local evidence of non-overlapping regions within $Q$ which have similar regions in $ref$ for obtaining global evidence on the hypothesis that $Q$ was generated from $ref$. Thus, for example, if we found 5 regions within $Q$ with similar copies in $ref$, each resulting with probability less than 10% of being generated by random, then the probability that $Q$ was generated by random is less than $(10\%)^5 = 0.001\%$ (and this is *despite the unfavorable assumption* we made that the rest of $Q$ was generated by random).

So far the segmentation $S$ was assumed to be given, and we estimated $GES(Q, H_{ref}, S)$. In order to obtain the global evidence score of $Q$, we marginalize over all possible segmentations $S$ of $Q$:

$$GES(Q|H_{ref}) = \log \frac{P(Q|H_{ref})}{P(Q|H_0)} = \log \sum_{S} P(S|H_{ref}) \frac{P(Q|H_{ref}, S)}{P(Q|H_0)} \tag{3}$$

Namely, the likelihood $P(S|H_{ref})$ of a segmentation $S$ can be interpreted as a weight for the likelihood ratio score of $Q$ induced by $S$. Thus, we would like $P(S|H_{ref})$ to reflect the complexity of the segmentation $S$ (e.g., its description length).

From a practical point of view, in most cases it would be intractable to compute $GES(Q|H_{ref})$, as Eq. (3) involves summation over all possible segmentations of the query $Q$. However, we can derive *upper and lower bounds* on $GES(Q|H_{ref})$ *which are easy to compute*:

**Claim 1.    Upper and lower bounds on GES:**

$$\max_{S} \{\ log P(S|H_{ref}) + \sum_{R_i \in S} LES(R_i|H_{ref}) \} \ \leq \ GES(Q|H_{ref}) \ \leq \ \sum_{q \in Q} PES(q|H_{ref})$$

$$\tag{4}$$

**proof:** *See Appendix* www.wisdom.weizmann.ac.il/~vision/Composition.html.

Practically, this claim implies that we do not need to scan all possible segmentations. The lower bound (left-hand side of Eq. (4) ) is achieved by the segmentation of $Q$ with the best accumulated evidence score, $\sum_{R_i \in S} LES(R_i|H_{ref}) = GES(Q|H_{ref}, S)$, penalized by the length of the segmentation description $log P(S|H_{ref}) = -length(S)$. Obviously, every segmentation provides such a lower (albeit less tight) bound on the total evidence score. Thus, if we find large enough contiguous regions in $Q$, with supporting regions in $ref$ (i.e., high enough local evidence scores), and define $R_0$ to be the remainder of $Q$, then $S = R_0, R_1, ..., R_k$ can provide a reasonable lower bound on $GES(Q|H_{ref})$. As to the upper bound on $GES(Q|H_{ref})$, this can be done by summing up the maximal point-wise evidence scores $PES(q|H_{ref})$ (see Eq. 2) from *all the points* in $Q$ (right-hand side of Eq. (4)). Note that the upper bound is computed by finding the maximal evidence regions that pass through every point in the query, regardless of the region complexity $length(R)$. Both bounds can be estimated quite efficiently (see Sec. 3).

## 3    Algorithmic Framework

The local and global evidence scores presented in Sec. 2 provide new local and global similarity measures for signal data, which can be used for various learning and inference problems (see Sec. 4). In this section we briefly describe the algorithmic framework used for computing $PES$, $LES$, and $GES$ to obtain the local and global compositional similarity measures.

Assume we are given a large region $R \subset Q$ and would like to estimate its evidence score $LES(R|H_{ref})$. We would like to find similar regions to $R$ in $ref$, that would provide large local evidence for $R$. However, (i) we cannot expect $R$ to appear as is, and would therefore like to allow for global and local deformations of $R$, and (ii) we would like to perform this search efficiently. Both requirements can be achieved by breaking $R$ into lots of small (partially overlapping) data patches, each with its own patch descriptor. This information is maintained via a geometric "ensemble" of local patch descriptors. The search for a similar ensemble in $ref$ is done using efficient inference on a star graphical model, while allowing for small local displacement of each local patch [5]. For example, in images these would be small spatial patches around each pixel contained in the larger image region $R$, and the displacements would be small shifts in $x$ and $y$. In video data the region $R$ would be a space-time volumetric region, and it would be broken into lots of small overlapping space-time volumetric patches. The local displacements would be in $x$, $y$, and $t$ (time). In audio these patches would be short time-frame windows, etc. In general, for any n-dimensional signal representation, the region $R$ would be a large n-dimensional region within the signal, and the patches would be small n-dimensional overlapping regions within $R$. The local patch descriptors are signal and application dependent, but can be very simple. (For example, in images we used a SIFT-like [9] patch descriptor computed in each image-patch. See more details in Sec. 4). It is the simultaneous matching of all these simple local patch descriptors with their relative positions that provides the strong overall evidence score for the entire region $R$. The likelihood of $R$, given a global transformation $T$ (e.g., location in $ref$) and local patch displacements $\Delta l_i$ for each patch $i$ in $R$ ($i = 1, 2, ..., |R|$), is captured by the following expression: $P(R|T, \{\Delta l_i\}, H_{ref}) = 1/Z \prod_i e^{-\frac{|\Delta d_i|^2}{2\sigma_1^2}} e^{-\frac{|\Delta l_i|^2}{2\sigma_2^2}}$ , where $\{\Delta d_i\}$ are the descriptor distortions of each patch, and $Z$ is a normalization factor. To estimate $P(R|T, H_{ref})$ we marginalize over all possible local displacements $\{\Delta l_i\}$ within a predefined limited radius. In order to compute $LES(R|H_{ref})$ in Eq. (1), we need to marginalize over all possible global transformations $T$. In our current implementation we used only global shifts, and assumed uniform distributions over all shifts, i.e., $P(T|H_{ref}) = 1/|ref|$. However, the algorithm can accommodate more complex global transformations. To compute $P(R|H_{ref})$, we used our inference algorithm described in [5], modified to compute likelihood (sum-product) instead of MAP (max-product). In a nutshell, the algorithm uses a few patches in $R$ (e.g., 2-3), exhaustively searching $ref$ for those patches. These patches restrict the possible locations of $R$ in $ref$, i.e., the possible candidate transformations $T$ for estimating $P(R|T, H_{ref})$. The search of each new patch is restricted to locations induced by the current list of candidate transformations $T$. Each new patch further reduces this list of candidate positions of $R$ in $ref$. This computation of $P(R|H_{ref})$ is efficient: $O(|db|) + O(|R|) \approx O(|db|)$, i.e., approximately linear in the size of $ref$.

In practice, we are not given a specific region $R \subset Q$ in advance. For each point $q \in Q$ we want to estimate its maximal region $R_{[q]}$ and its corresponding evidence score $LES(R|H_{ref})$ (Sec. 2.1). In

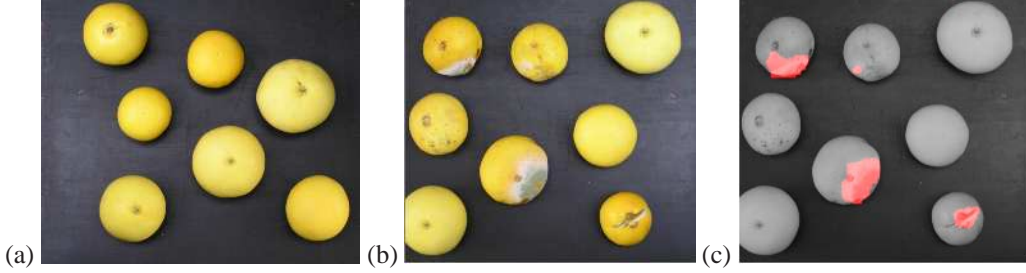

(a)&emsp;&emsp;&emsp;&emsp;&emsp;&emsp;(b)&emsp;&emsp;&emsp;&emsp;&emsp;&emsp;(c)

Figure 2: **Detection of defects in grapefruit images.** *Using the single image (a) as a "reference" of good quality grapefruits, we can detect defects (irregularities) in an image (b) of different grapefruits at different arrangements. Detected defects are highlighted in red (c).*

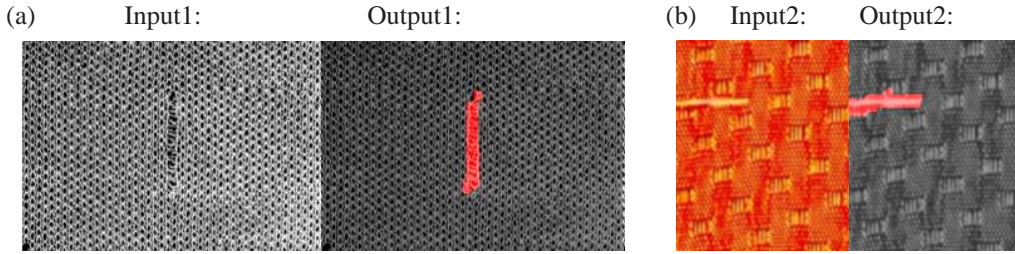

(a)&emsp;&emsp;Input1:&emsp;&emsp;&emsp;&emsp;Output1:&emsp;&emsp;&emsp;&emsp;(b)&emsp;&emsp;Input2:&emsp;&emsp;&emsp;Output2:

Figure 3: **Detecting defects in fabric images (No prior examples).** *Left side of (a) and (b) show fabrics with defects. Right side of (a) and (b) show detected defects in red (points with small intra-image evidence LES). Irregularities are measured relative to other parts of each image.*

order to perform this step efficiently, we start with a small surrounding region around $q$, break it into patches, and search only for that region in $ref$ (using the same efficient search method described above). Locations in $ref$ where good initial matches were found are treated as candidates, and are gradually 'grown' to their maximal possible matching regions (allowing for local distortions in patch position and descriptor, as before). The evidence score $LES$ of each such maximally grown region is computed. Using all these maximally grown regions we approximate $PES(q|H_{ref})$ and $R_{[q]}$ (for all $q \in Q$). In practice, a region found maximal for one point is likely to be the maximal region for many other points in $Q$. Thus the number of different maximal regions in $Q$ will tend to be *significantly smaller* than the number of points in $Q$.

Having computed $PES(q|H_{ref})$ $\forall q \in Q$, it is straightforward to obtain an upper bound on $GES(Q|H_{ref})$ (right-hand side of Eq. (4)). In principle, in order to obtain a lower bound on $GES(Q|H_{ref})$ we need to perform an optimization over all possible segmentations $S$ of $Q$. However, any good segmentation can be used to provide a reasonable (although less tight) lower bound. Having extracted a list of disjoint maximal regions $R_1, ..., R_k$, we can use these to induce a reasonable (although not optimal) segmentation using the following heuristics: We choose the first segment to be the maximal region with the largest evidence score: $\tilde{R}_1 = argmax_{R_i} LES(R_i|H_{ref})$. The second segment is chosen to be the largest of all the remaining regions after having removed their overlap with $\tilde{R}_1$, etc. This process yields a segmentation of $Q$: $S = \{\tilde{R}_1, ..., \tilde{R}_l\}$ ($l \leq k$). Re-evaluating the evidence scores $LES(\tilde{R}_i|H_{ref})$ of these regions, we can obtain a reasonable lower bound on $GES(Q|H_{ref})$ using the left-hand side of Eq. (4). For evaluating the lower bound, we also need to estimate $\log P(S|H_{ref}) = -length(S|H_{ref})$. This is done by summing the description length of the boundaries of the individual regions within $S$. For more details see appendix in www.wisdom.weizmann.ac.il/~vision/Composition.html.

## 4&emsp;Applications and Results

The *global similarity measure* $GES(Q|H_{ref})$ can be applied between individual signals, and/or between groups of signals (by setting $Q$ and $ref$ accordingly). As such it can be employed in machine learning tasks like retrieval, classification, recognition, and clustering. The *local similarity measure* $LES(R|H_{ref})$ can be used for local inference problems, such as local classification, saliency, segmentation, etc. For example, the local similarity measure can also be applied *between different*

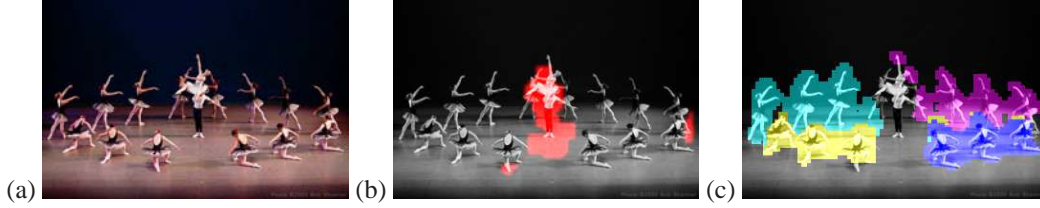

(a)            (b)            (c)

Figure 4: **Image Saliency and Segmentation.** *(a) Input image. (b) Detected salient points, i.e., points with low intra-image evidence scores LES (when measured relative to the rest of the image). (c) Image segmentation – results of clustering all the non-salient points into 4 clusters using normalized cuts. Each maximal region $R_{[q]}$ provides high evidence (translated to high affinity scores) that all the points within it should be grouped together (see text for more details).*

*portions of the same signal* (e.g., by setting $Q$ to be one part of the signal, and $ref$ to be the rest of the signal). Such intra-signal evidence can be used for inference tasks like segmentation, while the *absence of intra-signal evidence* (local **dis**similarity) can be used for detecting saliency/irregularities. In this section we demonstrate the applicability of our measures to several of these problems, and apply them to three different types of signals: audio, images, video. For additional results as well as video sequences see `www.wisdom.weizmann.ac.il/~vision/Composition.html`

**1. Detection of Saliency/Irregularities** *(in Images)***:** Using our statistical framework, we define a point $q \in Q$ to be irregular if its best local evidence score $LES(R_{[q]}|H_{ref})$ is below some threshold. Irregularities can be inferred either relative to a database of examples, or relative to the signal itself. In Fig. 2 we show an example of applying this approach for detecting defects in fruit. Using a single image as a "reference" of good quality grapefruits (Fig. 2.a, used as $ref$), we can detect defects (irregularities) in an image of different grapefruits at different arrangements (Fig. 2.b, used as the query $Q$). The algorithm tried to compose $Q$ from as large as possible pieces of $ref$. Points in $Q$ with low $LES$ (i.e., points whose maximal regions were small) were determined as irregular. These are highlighted in "red" in Fig. 2.c, and correspond to defects in the fruit.

Alternatively, local saliency within a query signal $Q$ can also be measured relative to other portions of $Q$, e.g., by trying to compose each region in $Q$ using pieces from the rest of $Q$. For each point $q \in Q$ we compute its *intra-signal evidence score $LES(R_{[q]})$* relative to the other (non-neighboring) parts of the image. Points with low intra-signal evidence are detected as salient. Examples of using intra-signal saliency to detect defects in fabric can be found in Fig. 3. Another example of using the same algorithm, but for a completely different scenario (a ballet scene) can be found in Fig. 4.b. We used a SIFT-like [9] patch descriptor, but computed densely for all local patches in the image. Points with low gradients were excluded from the inference (e.g., floor).

**2. Signal Segmentation** *(Images)***:** For each point $q \in Q$ we compute its maximal evidence region $R_{[q]}$. This can be done either relative to a different reference signal, or relative $Q$ itself (as is the case of saliency). Every maximal region provides evidence to the fact that all points within the region should be clustered/segmented together. Therefore, the value $LES(R_{[q]}|H_{ref}))$ is added to all entries $(i,j)$ in an affinity matrix, $\forall q_i \forall q_j \in R_{[q]}$. Spectral clustering can then be applied to the affinity matrix. Thus, large regions which appear also in $ref$ (in the case of a single image – other regions in $Q$) are likely to be clustered together in $Q$. This way we foster the generation of segments based on high evidential co-occurrence in the examples rather than based on low level similarity as in [10]. An example of using this algorithm for image segmentation is shown in Fig. 4.c. Note that we have not used explicitly low level similarity in neighboring point, as is customary in most image segmentation algorithms. Such additional information would improve the segmentation results.

**3. Signal Classification** *(Video – Action Classification)***:** We have used the action video database of [4], which contains different types of actions ("run", "walk", "jumping-jack", "jump-forward-on-two-legs", "jump-in-place-on-two-legs", "gallop-sideways", "wave-hand(s)","bend") performed by nine different people (altogether 81 video sequences). We used a leave-one-out procedure for action classification. The number of correct classifications was $79/81 = 97.5\%$. These sequences contain a single person in the field of view (e.g., see Fig. 5.a). Our method can handle much more complex scenarios. To illustrate the capabilities of our method we further added a few more sequences (e.g., see Fig. 5.b and 5.c), where several people appear simultaneously in the field of view, with partial

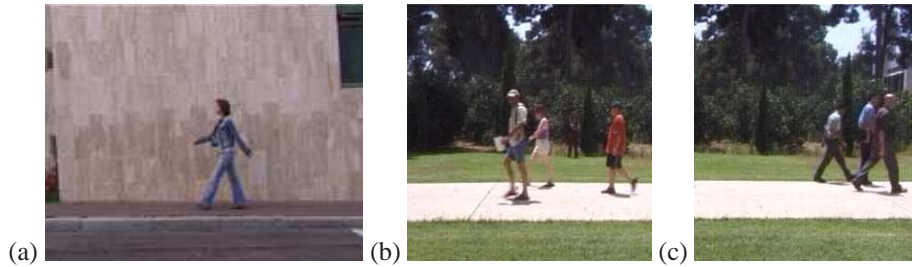

<div style="text-align:center">Figure 5: <strong>Action Classification in Video.</strong></div>

*(a) A sample 'walk' sequence from the action database of [4]. (b),(c) Other more complex sequences with several walking people in the field of view. Despite partial occlusions, differences in scale, and complex backgrounds, these sequences were all classified correctly as 'walk' sequences. For video sequences see* `www.wisdom.weizmann.ac.il/˜vision/Composition.html`

occlusions, some differences in scale, and more complex backgrounds. The complex sequences were all correctly classified (increasing the classification rate to $98\%$).

In our implementation, 3D space-time video regions were broken into small spatio-temporal video patches ($7 \times 7 \times 4$). The descriptor for each patch was a vector containing the absolute values of the temporal derivatives in all pixels of the patch, normalized to a unit length. Since stationary backgrounds have zero temporal derivatives, our method is not sensitive to the background, nor does it require foreground/background separation.

Image patches and fragments have been employed in the task of class-based object recognition (e.g., [7, 2, 6]). A sparse set of informative fragments were learned for a large class of objects (the training set). These approaches are useful for recognition, but are not applicable to non-class based inference problems (such as similarity between pairs of signals with no prior data, clustering, etc.)

**4. Signal Retrieval** *(Audio – Speaker Recognition)*: We used a database of 31 speakers (male and female). All the speakers repeated three times a five-word sentence (2-3 seconds long) in a foreign language, recorded over a phone line. Different repetitions by the same person slightly varied from one another. Altogether the database contained 93 samples of the sentence. Such short speech signals are likely to pose a problem for learning-based (e.g., HMM, GMM) recognition system. We applied our global measure $GES$ for retrieving the closest database elements. The highest $GES$ recognized the right speaker 90 out of 93 cases (i.e., $97\%$ correct recognition). Moreover, the second best $GES$ was correct 82 out of 93 cases ($88\%$). We used a standard mel-frequency cepstrum frame descriptors for time-frames of 25 msec, with overlaps of $50\%$.

**Acknowledgments**

Thanks to Y. Caspi, A. Rav-Acha, B. Nadler and R. Basri for their helpful remarks. This work was supported by the Israeli Science Foundation (Grant 281/06) and by the Alberto Moscona Fund. The research was conducted at the Moross Laboratory for Vision & Motor Control at the Weizmann Inst.

## References

[1] S. Altschul, W. Gish, W. Miller, E. Myers, and D. Lipman. Basic local alignment search tool. *JMolBiol*, 215:403–410, 1990.

[2] E. Bart and S. Ullman. Class-based matching of object parts. In *VideoRegister04*, page 173, 2004.

[3] A. Birnbaum. On the foundations of statistical inference. *J. Amer. Statist. Assoc*, 1962.

[4] M. Blank, L. Gorelick, E. Shechtman, M. Irani, and R. Basri. Actions as space-time shapes. In *ICCV05*.

[5] O. Boiman and M. Irani. Detecting irregularities in images and in video. In *ICCV05*, pages I: 462–469.

[6] P. Felzenszwalb and D. Huttenlocher. Pictorial structures for object recognition. *IJCV*, 61, 2005.

[7] R. Fergus, P. Perona, and A. Zisserman. Object class recognition by unsupervised scale-invariant learning. In *CVPR03*.

[8] S. Lazebnik, C. Schmid, and J. Ponce. Beyond bags of features: Spatial pyramid matching for recognizing natural scene categories. In *CVPR06*.

[9] D. Lowe. Distinctive image features from scale-invariant keypoints. *IJCV*, 60(2):91–110, 2004.

[10] J. Shi and J. Malik. Normalized cuts and image segmentation. *PAMI*, 22(8):888–905, August 2000.

[11] J. Sivic, B. Russell, A. Efros, A. Zisserman, and W. Freeman. Discovering objects and their localization in images. In *ICCV05*, pages I: 370–377.

[12] P. Viola and W. Wells, III. Alignment by maximization of mutual information. In *ICCV95*, pages 16–23.
